# Learning to Perceive Transparency from the Statistics of Natural Scenes

**Anat Levin**      **Assaf Zomet**      **Yair Weiss**
School of Computer Science and Engineering
The Hebrew University of Jerusalem
91904 Jerusalem, Israel
*{alevin,zomet,yweiss}@cs.huji.ac.il*

## Abstract

Certain simple images are known to trigger a percept of transparency: the input image $I$ is perceived as the sum of two images $I(x, y) = I_1(x, y) + I_2(x, y)$. This percept is puzzling. First, why do we choose the "more complicated" description with two images rather than the "simpler" explanation $I(x, y) = I_1(x, y) + 0$ ? Second, given the infinite number of ways to express $I$ as a sum of two images, how do we compute the "best" decomposition ?

Here we suggest that transparency is the rational percept of a system that is adapted to the statistics of natural scenes. We present a probabilistic model of images based on the qualitative statistics of derivative filters and "corner detectors" in natural scenes and use this model to find the most probable decomposition of a novel image. The optimization is performed using loopy belief propagation. We show that our model computes perceptually "correct" decompositions on synthetic images and discuss its application to real images.

## 1   Introduction

Figure 1a shows a simple image that evokes the percept of transparency. The image is typically perceived as a superposition of two layers: either a light square with a dark semitransparent square in front of it or a dark square with a light semitransparent square in front of it.

Mathematically, our visual system is taking a single image $I(x, y)$ and representing as the sum of two images:

$$I_1(x, y) + I_2(x, y) = I(x, y) \tag{1}$$

When phrased this way, the decomposition is surprising. There are obviously an infinite number of solutions to equation 1, how does our visual system choose one? Why doesn't our visual system prefer the "simplest" explanation $I(x, y) = I_1(x, y) + 0$ ?

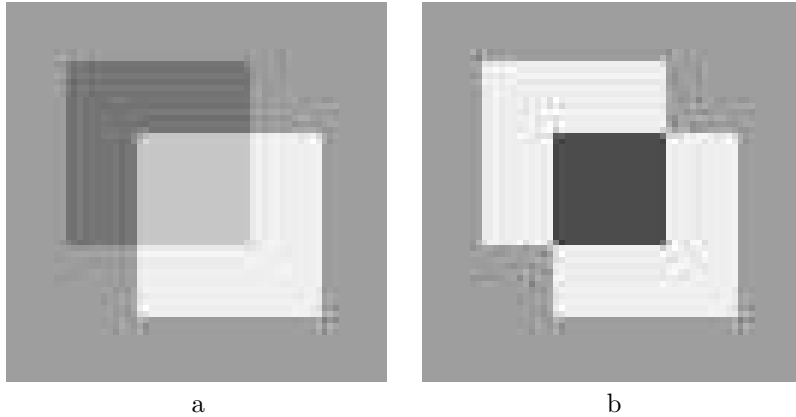

a                                    b

Figure 1: **a.** A simple image that evokes the percept of transparency. **b.** A simple image that does not evoke the percept of transparency.

Figure 1b shows a similar image that does not evoke the percept of transparency. Here again there are an infinite number of solutions to equation 1 but our visual system prefers the single layer solution.

Studies of the conditions for the percept of transparency go back to the very first research on visual perception (see [1] and references within). Research of this type has made great progress in understanding the types of junctions and their effects (e.g. X junctions of a certain type trigger transparency, T junctions do not). However, it is not clear how to apply these rules to an arbitrary image.

In this paper we take a simple Bayesian approach. While equation 1 has an infinite number of possible solutions, if we have prior probabilities $P(I_1(x,y)), P(I_2(x,y))$ then some of these solutions will be more probable than others. We use the statistics of natural images to define simple priors and finally use loopy belief propagation to find the most probable decomposition. We show that while the model knows nothing about "T junctions" or "X junctions", it can generate perceptually correct decompositions from a single image.

## 2    Statistics of natural images

A remarkably robust property of natural images that has received much attention lately is the fact that when derivative filters are applied to natural images, the filter outputs tend to be sparse [5, 7]. Figure 2 illustrates this fact: the histogram of the horizontal derivative filter is peaked at zero and fall off much faster than a Gaussian. Similar histograms are observed for vertical derivative filters and for the gradient magnitude: $|\nabla I|$.

There are many ways to describe the non Gaussian nature of this distribution (e.g. high kurtosis, heavy tails). Figure 2b illustrates the observation made by Mallat [4] and Simoncelli [8]: that the distribution is similar to an exponential density with exponent less than 1. We show the log probability for densities of the form $p(x) \propto e^{-x^{\alpha}}$. We assume $x \in [0, 100]$ and plot the log probabilities so that they agree on $p(0), p(100)$. There is a qualitative difference between distributions for which $\alpha > 1$ (when the log probability is convex) and those for which $\alpha < 1$ (when it becomes concave). As figure 2d shows, the natural statistics for derivative

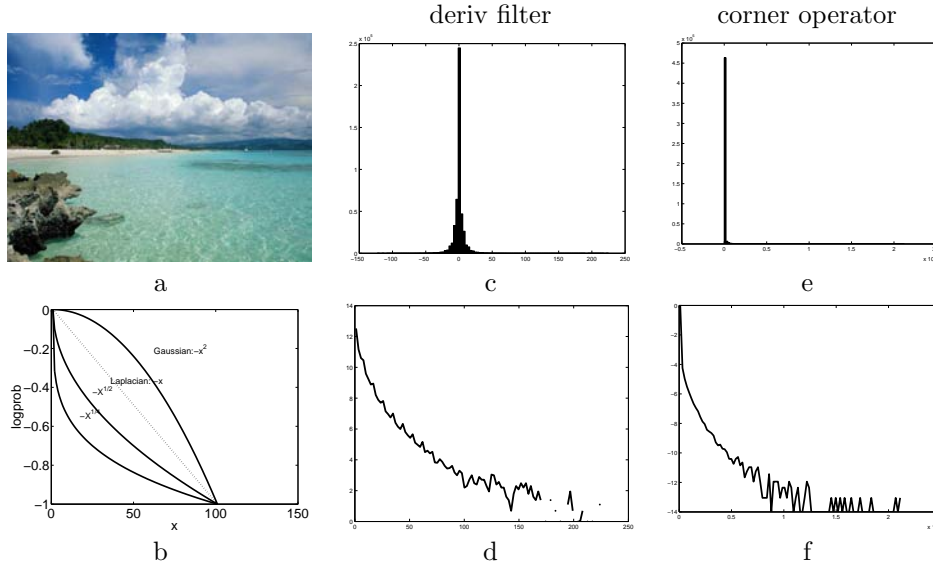

Figure 2: **a.** A natural image. **c** Histogram of filter outputs. **e** Histogram of corner detector outputs. **d,e** log histograms.

filters has the qualitative nature of a distribution $e^{-x^\alpha}$ with $\alpha < 1$.

In [9] the sparsity of derivative filters was used to decompose an image sequence as a sum of two image sequences. Will this prior be sufficient for a single frame ? Note that decomposing the image in figure 1a into two layers does not change the output of derivative filters: exactly the same derivatives exist in the single layer solution as in the two layer solution. Thus we cannot appeal to the marginal histogram of derivative filters to explain the percept of transparency in this image.

There are two ways to go beyond marginal histograms of derivative filters. We can either look at joint statistics of derivative filters at different locations or orientations [6] or look at marginal statistics of more complicated feature detectors (e.g. [11]).

We looked at the marginal statistics of a "corner detector". The output of the "corner detector" at a given location $x_0, y_0$ is defined as:

$$c(x_0, y_0) = det(\sum w(x, y) \begin{pmatrix} I_x^2(x, y) & I_x(x, y)I_y(x, y) \\ I_x(x, y)I_y(x, y) & I_y^2(x, y) \end{pmatrix}) \tag{2}$$

where $w(x, y)$ is a small Gaussian window around $x_0, y_0$ and $I_x, I_y$ are the derivatives of the image.

Figures 2e,f show the histogram of this corner operator on a typical natural image. Again, note that it has the qualitative statistic of a distribution $e^{-x^\alpha}$ for $\alpha < 1$.

To get a more quantitative description of the statistics we used maximum likelihood to fit a distribution of the form $P(x) = \frac{1}{Z}e^{-ax^\alpha}$ to gradient magnitudes and corner detector histograms in a number of images. We found that the histograms shown in figure 2 are typical: for both gradients and corner detectors the exponent was less than 1 and the exponent for the corner detector was smaller than that of the gradients. Typical exponents were 0.7 for the derivative filter and 0.25 for the corner detector. The scaling parameter $a$ of the corner detector was typically larger than

that of the gradient magnitude.

# 3 Simple prior predicts transparency

Motivated by the qualitative statistics observed in natural images we now define a probability distribution over images. We define the log probability of an image by means of a probability over its gradients:

$$\log P(I_x, I_y) = -\log Z - \sum_{x,y} \left( |\nabla I(x,y)|^\alpha + \eta c(x,y)^\beta \right) \qquad (3)$$

with $\alpha = 0.7, \beta = 0.25$. The parameter $\eta$ was determined by the ratio of the scaling parameters in the corner and gradient distributions.

Given a candidate decomposition of an image $I$ into $I_1$ and $I_2 = I - I_1$ we define the log probability of the decomposition as the sum of the log probabilities of the gradients of $I_1$ and $I_2$. Of course this is only an approximation: we are ignoring dependencies between the gradients across space and orientation. Although this is a weak prior, one can ask: is this enough to predict transparency? That is, is the most probable interpretation of figure 1a one with two layers and the most probable decomposition of figure 1b one with a single layer?

Answering this question requires finding the global maximum of equation 3. To gain some intuition we calculated the log probability of a one dimensional family of solutions. We defined $s(x,y)$ the image of a single white square in the same location as the bottom right square in figure 1a,b. We considered decompositions of the form $I_1 = \gamma s(x,y), I_2 = I - I_1$ and evaluated the log probability for values of $\gamma$ between $-1$ and $2$.

Figure 3a shows the result for figure 1a. The most probable decomposition is the one that agrees with the percept: $\gamma = 1$ one layer for the white square and another for the gray square. Figure 3b shows the result for figure 1b. The most probable decomposition again agrees with the percept: $\gamma = 0$ so that one layer is zero and the second contains the full image.

## 3.1 The importance of being non Gaussian

Equation 3 can be verbally described as preferring decompositions where the total edge and corner detector magnitudes are minimal. Would any cost function that has this preference give the same result?

Figure 3c shows the result with $\alpha = \beta = 2$ for the transparency figure (figure 1a). This would be the optimal interpretation if the marginal histograms of edge and corner detectors were Gaussian. Now the optimal interpretation indeed contains two layers but they are not the ones that humans perceive. Thus the non Gaussian nature of the histograms is crucial for getting the transparency percept. Similar "non perceptual" decompositions are obtained with other values of $\alpha, \beta > 1$.

We can get some intuition for the importance of having exponents smaller than 1 from the following observation which considers the analog of the transparency problem with scalars. We wish to solve the equation $a + b = 1$ and we have a prior over positive scalars of the form $P(x)$.

*Observation:* The MAP solution to the scalar transparency problem is obtained with $a = 1, b = 0$ or $a = 0, b = 1$ if and only if $\log P(x)$ is concave.

The proof follows directly from the definition of concavity.

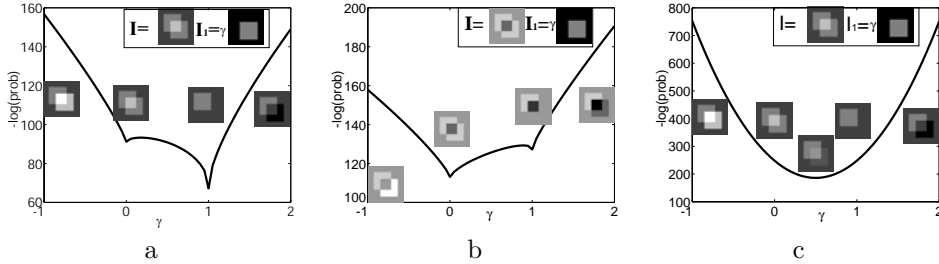

Figure 3: **a-b.** negative log probability (equation 3) for a sequence of decompositions of figure 1a,b respectively. The first layer is always a single square with contrast $\gamma$ and the second layer is shown in the insets. **c.** negative log probability (equation 3) for a sequence of decompositions of figure 1a with $\alpha = \beta = 2$.

## 4   Optimization using loopy BP

Finding the most likely decomposition requires a highly nonlinear optimization. We chose to discretize the problem and use max-product loopy belief propagation to find the optimum. We defined a graphical model in which every node $g_i$ corresponded to a discretization of the gradient of one layer $I_1$ at that location $g_i = (g_{ix}, g_{iy})^T$. For every value of $g_i$ we defined $f_i$ which represents the gradient of the second layer at that location: $f_i = (I_x, I_y)^T - g_i$. Thus the two gradients fields $\{g_i\}, \{f_i\}$ represent a valid decomposition of the input image $I$.

The joint probability is given by:

$$P(g) = \frac{1}{Z} \prod_i \Psi_i(g_i) \prod_{<ijkl>} \Psi_{ijkl}(g_i, g_j, g_k, g_l) \tag{4}$$

where $< ijkl >$ refers to four adjacent pixels that form a $2x2$ local square.

The local potential $\Psi_i(g_i)$ is based on the histograms of derivative filters:

$$\Psi_i(g_i) = e^{(-|g|^\alpha - |f|^\alpha)/T} \tag{5}$$

where $T$ is an arbitrary system "temperature".

The fourway potential: $\Psi_{ijkl}(g_i, g_j, g_k, g_l)$ is based on the histogram of the corner operator:

$$\Psi_{ijkl}(g_i, g_j, g_k, g_l) = e^{-\eta/T\left(det(g_i g_i^T + g_j g_j^T + g_k g_k^T + g_l g_l^T)^\beta + det(f_i f_i^T + f_j f_j^T + f_k f_k^T + f_l f_l^T)^\beta\right)} \tag{6}$$

To enforce integrability of the gradient fields the fourway potential is set to zero when $g_i, g_j, g_k, g_l$ violate the integrability constraint (cf. [3]).

The graphical model defined by equation 4 has many loops. Nevertheless motivated by the recent results on similar graphs [2, 3] we ran the max-product belief propagation algorithm on it. The max-product algorithm finds a gradient field $\{g_i\}$ that is a local maximum of equation 4 with respect to a large neighbourhood [10]. This gradient field also defines the complementary gradient field $\{f_i\}$ and finally we integrate the two gradient fields to find the two layers. Since equation 4 is completely symmetric in $\{f\}$ and $\{g\}$ we break the symmetry by requiring that the gradient in a *single location* $g_{i_0}$ belong to layer 1.

In order to run BP we need to somehow discretize the space of possible gradients at each pixel. Similar to the approach taken in [2] we use the local potentials to

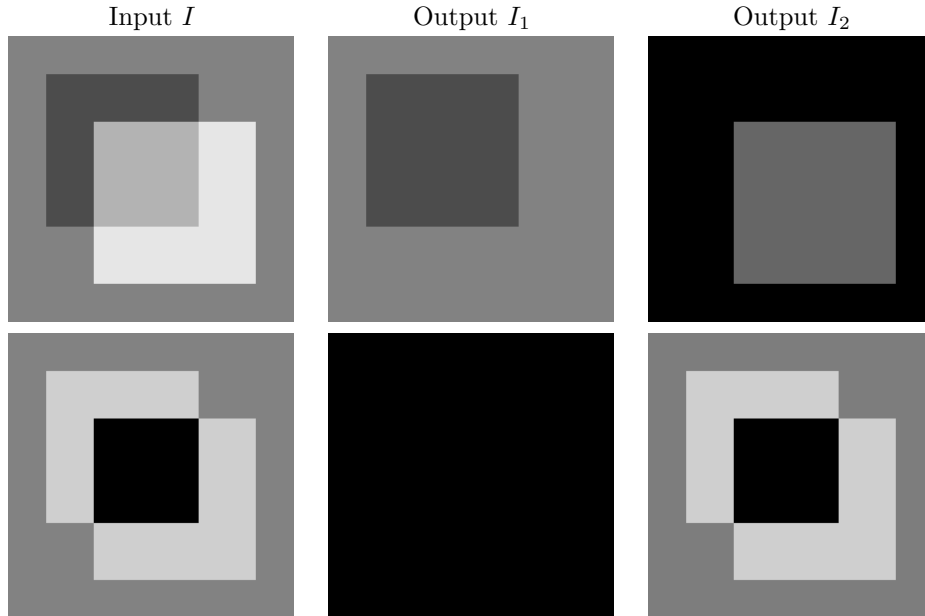

Figure 4: Output of the algorithm on synthetic images. The algorithm effectively searches over an exponentially large number of possible decompositions and chooses decompositions that agree with the percept.

sample a small number of candidate gradients at each pixel. Since the local potential penalizes non zero gradients, the most probable candidates are $g_i = (I_x, I_y)$ and $g_i = (0,0)$. We also added two more candidates at each pixel $g_i = (I_x, 0)$ and $g_i = (0, I_y)$. With this discretization there are still an exponential number of possible decompositions of the image. We have found that the results are unchanged when more candidates are introduced at each pixel.

Figure 4 shows the output of the algorithm on the two images in figure 1. An animation that illustrates the dynamics of BP on these images is available at www.cs.huji.ac.il/ ~yweiss. Note that the algorithm is essentially searching exponentially many decompositions of the input images and knows nothing about "X junctions" or "T junctions" or squares. Yet it finds the decompositions that are consistent with the human percept.

Will our simple prior also allow us to decompose a sum of two real images ? We first tried a one dimensional family of solutions as in figure 3. We found that for real images that have very little texture (e.g. figure 5b) the maximal probability solution is indeed obtained at the perceptually correct solution. However, nearly any other image that we tried had some texture and on such images the model failed (e.g. 5a). When there is texture in both layers, the model always prefers a one layer decomposition: the input image plus a zero image. To understand this failure, recall that the model prefers decompositions that have few corners and few edges. According to the simple "edge" and "corner" operators that we have used, real images have edges and corners at nearly every pixel so the two layer decomposition has twice as many edges and corners as the one layer decomposition. To decompose general real images we need to use more sophisticated features to define our prior.

Even for images with little texture standard belief propagation with synchronous

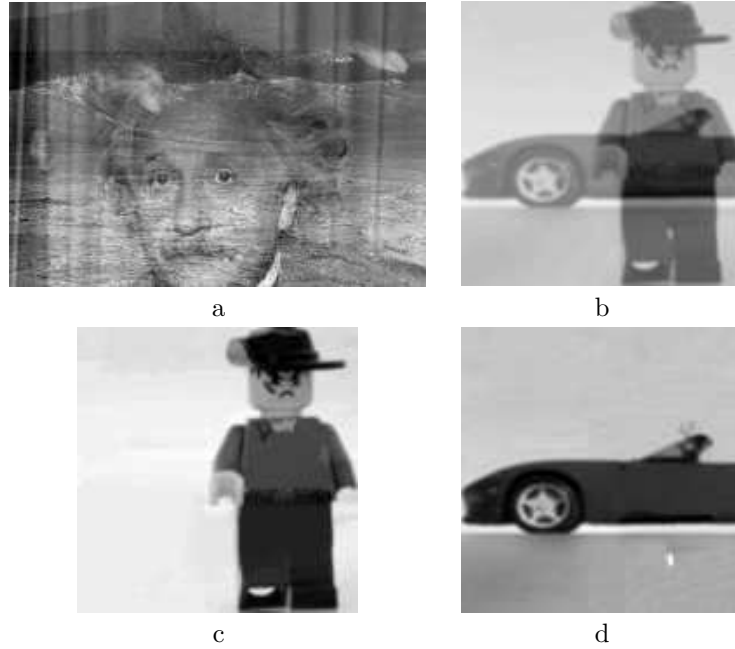

a                                    b

c                                    d

Figure 5: When we sum two arbitrary images (e.g. in **a.**) the model usually prefers the one layer solution. This is because of the texture that results in gradients and corners at every pixel. For real images that are relatively texture free (e.g. in **b.**) the model does prefer splitting into two layers (**c.** and **d.**)

updates did not converge. Significant manual tweaking was required to get BP to converge. First, we manually divided the input image into smaller patches and ran BP separately on each patch. Second, to minimize discretization artifacts we used a different number of gradient candidates at each pixel and always included the gradients of the original images in the list of candidates at that pixel. Third, to avoid giving too much weight to corners and edges in textured regions, we increased the temperature at pixels where the gradient magnitude was not a local maximum. The results are shown at the bottom of 5. In preliminary experiments we have found that similar results can be obtained with far less tweaking when we use generalized belief propagation to do the optimization.

## 5    Discussion

The percept of transparency is a paradigmatic example of the ill-posedness of vision: the number of equations is half the number of unknowns. Nevertheless our visual systems reliably and effectively compute a decomposition of a single image into two images. In this paper we have argued that this perceptual decomposition may correspond to the most probable decomposition using a simple prior over images derived from natural scene statistics.

We were surprised with the mileage we got out of the very simple prior we used: even though it only looks at two operators (gradients, and cornerness) it can generate surprisingly powerful predictions. However, our experiments with real images show that this simple prior is not powerful enough. In future work we would like to add additional features. One way to do this is by defining features that look for

"texture edges" and "texture corners" and measuring their statistics in real images. A second way to approach this is to use a full exponential family maximum likelihood algorithm (e.g. [11]) that automatically learned which operators to look at as well as the weights on the histograms.

## References

[1] E.H. Adelson. Lightness perception and lightness illusions. In M. Gazzaniga, editor, *The new cognitive neurosciences*, 2000.

[2] W.T. Freeman and E.C. Pasztor. Learning to estimate scenes from images. In M.S. Kearns, S.A. Solla, and D.A. Cohn, editors, *Adv. Neural Information Processing Systems 11*. MIT Press, 1999.

[3] B.J. Frey, R. Koetter, and N. Petrovic. Very loopy belief propagation for unwrapping phase images. In *Adv. Neural Information Processing Systems 14*. 2001.

[4] S. Mallat. A theory for multiresolution signal decomposition : the wavelet representation. *IEEE Trans. PAMI*, 11:674–693, 1989.

[5] B.A. Olshausen and D. J. Field. Emergence of simple-cell receptive field properties by learning a sparse code for natural images. *Nature*, 381:607–608, 1996.

[6] J. Portilla and E. P. Simoncelli. A parametric texture model based on joint statistics of complex wavelet coefficients. *Int'l J. Comput. Vision*, 40(1):49–71, 2000.

[7] E.P. Simoncelli. Statistical models for images:compression restoration and synthesis. In *Proc Asilomar Conference on Signals, Systems and Computers*, pages 673–678, 1997.

[8] E.P. Simoncelli. Bayesian denoising of visual images in the wavelet domain. In P Mller and B Vidakovic, editors, *Wavelet based models*, 1999.

[9] Y. Weiss. Deriving intrinsic images from image sequences. In *Proc. Intl. Conf. Computer Vision*, pages 68–75. 2001.

[10] Y. Weiss and W.T. Freeman. On the optimality of solutions of the max-product belief propagation algorithm in arbitrary graphs. *IEEE Transactions on Information Theory*, 47(2):723–735, 2001.

[11] Song Chun Zhu, Zing Nian Wu, and David Mumford. Minimax entropy principle and its application to texture modeling. *Neural Computation*, 9(8):1627–1660, 1997.
